# Mistake Bounds
# for Maximum Entropy Discrimination

**Philip M. Long**
Center for Computational Learning Systems
Columbia University
plong@cs.columbia.edu

**Xinyu Wu**
Department of Computer Science
National University of Singapore
wuxy@comp.nus.edu.sg

## Abstract

We establish a mistake bound for an ensemble method for classification based on maximizing the entropy of voting weights subject to margin constraints. The bound is the same as a general bound proved for the Weighted Majority Algorithm, and similar to bounds for other variants of Winnow. We prove a more refined bound that leads to a nearly optimal algorithm for learning disjunctions, again, based on the maximum entropy principle. We describe a simplification of the on-line maximum entropy method in which, after each iteration, the margin constraints are replaced with a single linear inequality. The simplified algorithm, which takes a similar form to Winnow, achieves the same mistake bounds.

## 1 Introduction

In this paper, we analyze a maximum-entropy procedure for ensemble learning in the on-line learning model. In this model, learning proceeds in trials. During the $t$th trial, the algorithm (1) receives $\mathbf{x}_t \in \{0, 1\}^n$ (interpreted in this work as a vector of base classifier predictions), (2) predicts a class $\hat{y}_t \in \{0, 1\}$, and (3) discovers the correct class $y_t$. During trial $t$, the algorithm has access only to information from previous trials.

The first algorithm we will analyze for this problem was proposed by Jaakkola, Meila and Jebara [14]. The algorithm, at each trial $t$, makes its prediction by taking a weighted vote over the predictions of the base classifiers. The weight vector $\mathbf{p}_t$ is the probability distribution over the $n$ base classifiers that maximizes the entropy, subject to the constraint that $\mathbf{p}_t$ correctly classifies all patterns seen in previous trials with a given margin $\gamma$. That is, it maximizes the entropy of $\mathbf{p}_t$ subject to the constraints that $\mathbf{p}_t \cdot \mathbf{x}_s \geq 1/2 + \gamma$ whenever $y_s = 1$ for $s < t$, and $\mathbf{p}_t \cdot \mathbf{x}_s \leq 1/2 - \gamma$ whenever $y_s = 0$ for $s < t$.

We show that, if there is a weighting $\mathbf{p}^*$, determined with benefit of hindsight, that achieves margin $\gamma$ on all trials, then this on-line maximum entropy procedure makes at most $\frac{\ln n}{2\gamma^2}$ mistakes.

Littlestone [19] proved the same bound for the Weighted Majority Algorithm [21], and a similar bound for the Balanced Winnow Algorithm [19]. The original Winnow algorithm was designed to solve the problem of learning a hidden disjunction of a small number $k$ out of a possible $n$ boolean variables. When this problem is reduced to our general setting in the most natural way, the resulting bound is $\Theta(k^2 \log n)$, whereas Littlestone

proved a bound of $ek \ln n$ for Winnow. We prove more refined bounds for a wider family of maximum-entropy algorithms, which use thresholds different than $1/2$ (as proposed in [14]) and class-sensitive margins. A mistake bound of $ek \ln n$ for learning disjunctions is a consequence of this more refined analysis.

The optimization needed at each round can be cast as minimizing a convex function subject to convex constraints, and thus can be solved in polynomial time [25]. However, the same mistake bounds hold for a similar, albeit linear-time, algorithm. This algorithm, after each trial, replaces all constraints from previous trials with a single linear inequality. (This is analogous to modification of SVMs leading to the ROMMA algorithm [18].) The resulting update is similar in form to Winnow.

Littlestone [19] analyzed some variants of Winnow by showing that mistakes cause a reduction in the relative entropy between the learning algorithm's weight vector, and that of the target function. Kivinen and Warmuth [16] showed that an algorithm related to Winnow trades optimally in a sense between accommodating the information from new data, and keeping the relative entropy between the new and old weight vectors small. Blum [4] identified a correspondence between Winnow and a different application of the maximum entropy principle, in which the algorithm seeks to maximize the average entropy of the conditional distribution over the class designations (the $y_t$'s) subject to constraints arising from the examples, as proposed in [2]. Our proofs have a similar structure to the analysis of ROMMA [18]. Our problems fall within the general framework analyzed by Gordon [11]; while Gordon's results expose interesting relationships among learning algorithms, applying them did not appear to be the most direct route to solving our concrete problem, nor did they appear likely to result in the most easily understood proofs. As in related analyses like mistake bounds for the perceptron algorithm [22], Winnow [19] and the Weighted Majority Algorithm [19], our bound holds for any sequence of $(\mathbf{x}_t, y_t)$ pairs satisfying the separation condition; in particular no independence assumptions are needed. Langford, Seeger and Megiddo [17] performed a related analysis, incomparable in strength, using independence assumptions. Other related papers include [3, 20, 5, 15, 26, 13, 8, 27, 7].

The proofs of our main results do not contain any calculation; they combine simple geometric arguments with established information theory. The proof of the main result proceeds roughly as follows. If there is a mistake on trial $t$, it is corrected with a large margin by $\mathbf{p}_{t+1}$. Thus $\mathbf{p}_{t+1}$ must assign a significantly different probability to the voters predicting 1 on trial $t$ than $\mathbf{p}_t$ does. Applying an identity known as Pinsker's inequality, this means that the relative entropy from $\mathbf{p}_{t+1}$ and $\mathbf{p}_t$ is large. Next, we exploit the fact that the constraints satisfied by $\mathbf{p}_t$, and therefore by $\mathbf{p}_{t+1}$, are convex to show that moving from $\mathbf{p}_t$ to $\mathbf{p}_{t+1}$ must take you *away* from the uniform distribution, thus decreasing the entropy. The theorem then follows from the fact that the entropy can only be reduced by a total of $\ln n$. The refinement leading to a $ek \ln n$ bound for disjunctions arises from the observation that Pinsker's inequality can be strengthened when the probabilities being compared are small.

The analysis of this paper lends support to a view of Winnow as a fast, incremental approximation to the maximum entropy discrimination approach, and suggests a variant of Winnow that corresponds more closely to the inductive bias of maximum entropy.

## 2  Preliminaries

Let $n$ be the number of base classifiers. To avoid clutter, for the rest of the paper, "probability distribution" should be understood to mean "probability distribution over $\{1, ..., n\}$."

## 2.1 Margins

For $u \in [0,1]$, define $\sigma(u) = 1$ to be 1 if $u \geq 1/2$, and 0 otherwise. For a feature vector $\mathbf{x} \in \{0,1\}^n$ and a class designation $y \in \{0,1\}$, say that a probability distribution $\mathbf{p}$ is *correct with margin* $\gamma$ if $\sigma(\mathbf{p} \cdot \mathbf{x}) = y$, and $|\mathbf{p} \cdot \mathbf{x} - 1/2| \geq \gamma$. If $\mathbf{x}$ and $y$ were encountered in a trial of a learning algorithm, we say that $\mathbf{p}$ is correct with margin $\gamma$ on that trial.

## 2.2 Entropy, relative entropy, and variation

Recall that, for a probability distributions $\mathbf{p} = (p_1, ..., p_n)$ and $\mathbf{q} = (q_1, ..., q_n)$,

- the *entropy* of $\mathbf{p}$, denoted by $H(\mathbf{p})$, is defined by $\sum_{i=1}^n p_i \ln(1/p_i)$,
- the *relative entropy* between $\mathbf{p}$ and $\mathbf{q}$, denoted by $D(\mathbf{p}||\mathbf{q})$, is defined by $\sum_{i=1}^n p_i \ln(p_i/q_i)$, and
- the *variation distance* between $\mathbf{p}$ and $\mathbf{q}$, denoted by $V(\mathbf{p}, \mathbf{q})$, is defined to be the maximum difference between the probabilities that they assign to any set:

$$V(\mathbf{p}, \mathbf{q}) = \max_{\mathbf{x} \in \{0,1\}^n} \mathbf{p} \cdot \mathbf{x} - \mathbf{q} \cdot \mathbf{x} = \frac{1}{2} \sum_{i=1}^n |p_i - q_i|. \qquad (1)$$

Relative entropy and variation distance are related in Pinsker's inequality.

**Lemma 1 ([23])** *For all $\mathbf{p}$ and $\mathbf{q}$, $D(\mathbf{p}||\mathbf{q}) \geq 2V(\mathbf{p}, \mathbf{q})^2$.*

## 2.3 Information geometry

Relative entropy obeys something like the Pythogarean Theorem.

**Lemma 2 ([9])** *Suppose $\mathbf{q}$ is a probability distribution, $C$ is a convex set of probability distributions, and $\mathbf{r}$ is the element of $A$ that minimizes $D(\mathbf{r}||\mathbf{q})$. Then for any $\mathbf{p} \in C$,*

$$D(\mathbf{p}||\mathbf{q}) \geq D(\mathbf{p}||\mathbf{r}) + D(\mathbf{r}||\mathbf{q}).$$

*If $C$ can be defined by a system of linear equations, then*

$$D(\mathbf{p}||\mathbf{q}) = D(\mathbf{p}||\mathbf{r}) + D(\mathbf{r}||\mathbf{q}).$$

# 3 Maximum Entropy with Margin

In this section, we will analyze the algorithm $\mathrm{OME}_\gamma$ ("on-line maximum entropy") that at the $t$th trial

- chooses $\mathbf{p}_t$ to maximize the entropy $H(\mathbf{p}_t)$, subject to the constraint that it is correct with margin $\gamma$ on all pairs $(\mathbf{x}_s, y_s)$ seen in the past (with $s < t$),
- predicts 1 if and only if $\mathbf{p}_t \cdot \mathbf{x}_t \geq 1/2$.

In our analysis, we will assume that there is always a feasible $\mathbf{p}_t$.

The following is our main result.

**Theorem 3** *If there is a fixed probability distribution $\mathbf{p}^*$ that is correct with margin $\gamma$ on all trials, $\mathrm{OME}_\gamma$ makes at most $\frac{\ln n}{2\gamma^2}$ mistakes.*

**Proof**: We will show that a mistake causes the entropy of the hypothesis to drop by at least $2\gamma^2$. Since the constraints only become more restrictive, the entropy never increases, and so the fact that the entropy lies between $0$ and $\ln n$ will complete the proof.

Suppose trial $t$ was a mistake. The definition of $\mathbf{p}_{t+1}$ ensures that $\mathbf{p}_{t+1} \cdot \mathbf{x}_t$ is on the correct side of $1/2$ by at least $\gamma$. But $\mathbf{p}_t \cdot \mathbf{x}_t$ was on the wrong side of $1/2$. Thus $|\mathbf{p}_{t+1} \cdot \mathbf{x}_t - \mathbf{p}_t \cdot \mathbf{x}_t| \geq \gamma$. Either $\mathbf{p}_{t+1} \cdot \mathbf{x}_t - \mathbf{p}_t \cdot \mathbf{x}_t \geq \gamma$, or the bitwise complement $c(\mathbf{x}_t)$ of $\mathbf{x}_t$ satisfies $\mathbf{p}_{t+1} \cdot c(\mathbf{x}_t) - \mathbf{p}_t \cdot c(\mathbf{x}_t) \geq \gamma$. Thus $V(\mathbf{p}_{t+1}, \mathbf{p}_t) \geq \gamma$. Therefore, Pinsker's Inequality (Lemma 1) implies that

$$D(\mathbf{p}_{t+1}||\mathbf{p}_t) \geq 2\gamma^2. \qquad (2)$$

Let $C_t$ be the set of all probability distributions that satisfy the constraints in effect when $\mathbf{p}_t$ was chosen, and let $\mathbf{u} = (1/n, ..., 1/n)$. Since $\mathbf{p}_{t+1}$ is in $C_t$ (it must satisfy the constraints that $\mathbf{p}_t$ did), Lemma 2 implies $D(\mathbf{p}_{t+1}||\mathbf{u}) \geq D(\mathbf{p}_{t+1}||\mathbf{p}_t) + D(\mathbf{p}_t||\mathbf{u})$ and thus $D(\mathbf{p}_{t+1}||\mathbf{u}) - D(\mathbf{p}_t||\mathbf{u}) \geq D(\mathbf{p}_{t+1}||\mathbf{p}_t)$ which, since $D(\mathbf{p}||\mathbf{u}) = (\ln n) - H(\mathbf{p})$ for all $\mathbf{p}$, implies $H(\mathbf{p}_t) - H(\mathbf{p}_{t+1}) \geq D(\mathbf{p}_{t+1}||\mathbf{p}_t)$. Applying (2), we get $H(\mathbf{p}_t) - H(\mathbf{p}_{t+1}) \geq 2\gamma^2$. As described above, this completes the proof. ∎

Because $H(\mathbf{p}_t)$ is always at least $H(\mathbf{p}^*)$, the same analysis leads to a mistake bound of $(\ln n - H(\mathbf{p}^*))/(2\gamma^2)$. Further, a nearly identical proof establishes the following (details are omitted from this abstract).

**Theorem 4** *Suppose* $\mathrm{OME}_\gamma$ *is modified so that* $\mathbf{p}_1$ *is set to be something other than the uniform distribution, and each* $\mathbf{p}_t$ *minimizes* $D(\mathbf{p}_t||\mathbf{p}_1)$ *subject to the same constraints.*

*If there is a fixed* $\mathbf{p}^*$ *that is correct with margin* $\gamma$ *on all trials, the modified algorithm makes at most* $\frac{D(\mathbf{p}^*||\mathbf{p}_1)}{2\gamma^2}$ *mistakes.*

## 4   Maximum Entropy for Learning Disjunctions

In this section, we show how the maximum entropy principle can be used to efficiently learn disjunctions.

For a threshold $b$, define $\sigma_b(x)$ to be $1$ if $x \geq b$ and $0$ otherwise. For a feature vector $\mathbf{x} \in \{0, 1\}^n$ and a class designation $y \in \{0, 1\}$, say that $\mathbf{p}$ is correct at threshold $b$ with margin $\gamma$ if $\sigma_b(\mathbf{p} \cdot \mathbf{x}) = y$, and $|\mathbf{p} \cdot \mathbf{x} - b| \geq \gamma$.

The algorithm $\mathrm{OME}_{b,\gamma_+,\gamma_-}$ analyzed in this section, on the $t$th trial

- chooses $\mathbf{p}_t$ to maximize the entropy $H(\mathbf{p}_t)$, subject to the constraint that it is correct at threshold $b$ with margin $\gamma_+$ on all pairs $(\mathbf{x}_s, y_s)$ with $y_s = 1$ seen in the past (with $s < t$), and correct at threshold $b$ with margin $\gamma_-$ on all such pairs $(\mathbf{x}_s, y_s)$ with $y_s = 0$, then
- predicts $1$ if and only if $\mathbf{p}_t \cdot \mathbf{x}_t \geq b$.

Note that the algorithm $\mathrm{OME}_\gamma$ considered in Section 3 can also be called $\mathrm{OME}_{1/2,\gamma,\gamma}$.

For $p, q \in [0, 1]$, define $d(p||q) = D((p, (1-p))||(q, (1-q)))$, often called "entropic loss."

**Lemma 5** *If there is an* $\mathbf{x} \in \{0, 1\}^n$ *such that* $\mathbf{p} \cdot \mathbf{x} = p$ *and* $\mathbf{q} \cdot \mathbf{x} = q$, *then* $D(\mathbf{p}||\mathbf{q}) \geq d(p||q)$.

**Proof**: Application of Lagrange multipliers, together with the fact that $D$ is convex [6], implies that $D(\mathbf{p}||\mathbf{q})$ is minimized, subject to the constraints that $\mathbf{p} \cdot \mathbf{x} = p$ and $\mathbf{q} \cdot \mathbf{x} = q$, when (1) $p_i$ is the same for all $i$ with $x_i = 1$, (2) $q_i$ is the same for all $i$ with $x_i = 1$, (3) $p_i$ is the same for all $i$ with $x_i = 0$, (4) $q_i$ is the same for all $i$ with $x_i = 0$. The

above four properties, together with the constraints, are enough to uniquely specify $\mathbf{p}$ and $\mathbf{q}$. Evaluating $D(\mathbf{p}||\mathbf{q})$ in this case gives the result. ∎

**Theorem 6** *Suppose there is a probability distribution $\mathbf{p}^*$ that is correct at threshold $b$, with a margin $\gamma_+$ on all trials $t$ with $y_t = 1$, and with margin $\gamma_-$ on all trials with $y_t = 0$. Then $\mathrm{OME}_{b,\gamma_+,\gamma_-}$ makes at most $\frac{\ln n}{\min\{d(b+\gamma_+||b),d(b-\gamma_-||b)\}}$ mistakes.*

**Proof**: The outline of the proof is similar to the proof of Theorem 3. We will show that mistakes cause the entropy of the algorithm's hypothesis to decrease.

Arguing as in the proof of Theorem 3, $H(\mathbf{p}_{t+1}) \leq H(\mathbf{p}_t) - D(\mathbf{p}_{t+1}||\mathbf{p}_t)$. Lemma 5 then implies that

$$H(\mathbf{p}_{t+1}) \leq H(\mathbf{p}_t) - d(\mathbf{p}_{t+1} \cdot \mathbf{x}_t||\mathbf{p}_t \cdot \mathbf{x}_t). \qquad (3)$$

If there was a mistake on trial $t$ for which $y_t = 1$, then $\mathbf{p}_t \cdot \mathbf{x}_t \leq b$, and $\mathbf{p}_{t+1} \cdot \mathbf{x}_t \geq b + \gamma_+$. Thus in this case $d(\mathbf{p}_{t+1} \cdot \mathbf{x}_t||\mathbf{p}_t \cdot \mathbf{x}_t) \geq d(b + \gamma_+||b)$. Similarly, if there was a mistake on trial $t$ for which $y_t = 0$, then $d(\mathbf{p}_{t+1} \cdot \mathbf{x}_t||\mathbf{p}_t \cdot \mathbf{x}_t) \geq d(b - \gamma_-||b)$.

Once again, these two bounds on $d(\mathbf{p}_{t+1} \cdot \mathbf{x}_t||\mathbf{p}_t \cdot \mathbf{x}_t)$, together with (3) and the fact that the entropy is between $0$ and $\ln n$, complete the proof. ∎

The analysis of Theorem 6 can also be used to prove bounds for the case in which mistakes of different types have different costs, as considered in [12].

Theorem 6 improves on Theorem 3 even in the case in which $\gamma_+ = \gamma_-$ and $b = 1/2$. For example, if $\gamma = 1/4$, Theorem 6 gives a bound of $7.65 \ln n$, where Theorem 3 gives an $8 \ln n$ bound.

Next, we apply Theorem 6 to analyze the problem of learning disjunctions.

**Corollary 7** *If there are $k$ of the $n$ features, such that each $y_t$ is the disjunction of those features in $\mathbf{x}_t$, then algorithm $\mathrm{OME}_{1/(ek),1/k-1/(ek),1/(ek)}$ makes at most $ek \ln n$ mistakes.*

**Proof Sketch**: If the target weight vector $\mathbf{p}^*$ assigns equal weight to each of the variables in the disjunction, when $y = 1$, the weight of variables evaluating to 1 is at least $1/k$, and when $y = 0$, it is 0. So the hypothesis of Theorem 6 is satisfied when $b = 1/(ek)$, $\gamma_+ = 1/k - b$ and $\gamma_- = b$. Plugging into Theorem 6, simplifying and overapproximating completes the proof. ∎

To get a more readable, but weaker, variant of Theorem 6, we will use the following bound, implicit in the analysis of Angluin and Valiant [1] (see Theorem 1.1 of [10] for a more explicit proof, and [24] for a closely related bound). It improves on Pinsker's inequality (Lemma 1) when $n = 2$, $p$ is small, and $q$ is close to $p$.

**Lemma 8 ([1])** *If $0 \leq p \leq 2q$, $d(p||q) \geq \frac{(p-q)^2}{3q}$.*

The following is a direct consequence of Lemma 8 and Theorem 6. Note that in the case of disjunctions, it leads to a weaker $6k \ln n$ bound.

**Theorem 9** *If there is a probability distribution $\mathbf{p}^*$ that is correct at threshold $b$ with a margin $\gamma$ on all trials, then $\mathrm{OME}_{b,\gamma,\gamma}$ makes at most $\frac{3b \ln n}{\gamma^2}$ mistakes.*

## 5 Relaxed on-line maximum entropy algorithms

Let us refer the halfspace of probability distributions that satisfy the constraint of trial $t$ as $T_t$ and the associated separating hyperplane by $J_t$. Recall that $C_t$ is the set of feasible

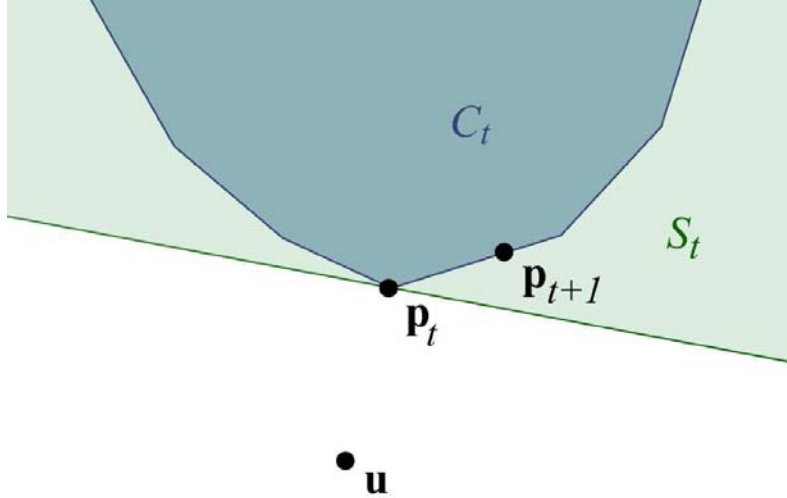

Figure 1: In ROME, the constraints $C_t$ in effect before the $t$th round are replaced by the halfspace $S_t$.

solutions to *all* the constraints in effect when $\mathbf{p}_t$ is chosen. So $\mathbf{p}_{t+1}$ maximizes entropy subject to membership in $C_{t+1} = T_t \cap C_t$.

Our proofs only used the following facts about the OME algorithm: (a) $\mathbf{p}_{t+1} \in T_t$, (b) $\mathbf{p}_t$ is the maximum entropy member of $C_t$, and (c) $\mathbf{p}_{t+1} \in C_t$.

Suppose $A_t$ is the set of weight vectors with entropy at last that of $\mathbf{p}_t$. Let $H_t$ be the hyperplane tangent to $A_t$ at $\mathbf{p}_t$. Finally, let $S_t$ be the halfspace with boundary $H_t$ containing $\mathbf{p}_{t+1}$. (See Figure 1.) Then (a), (b) and (c) hold if $C_t$ is replaced with $S_t$. (The least obvious is (b), which follows since $H_t$ is tangent to $A_t$ at $\mathbf{p}_t$, and the entropy function is strictly concave.)

Also, as previously observed by Littlestone [19], the algorithm might just as well not respond to trials in which there is not a mistake. Let us refer to an algorithm that does both of these as a Relaxed On-line Maximum Entropy (ROME) algorithm.

A similar observation regarding an on-line SVM algorithm, led to the simple ROMMA algorithm [18]. In that case, it was possible to obtain a simple close-form expression for the new weight vector. Matters are only slightly more complicated here.

**Proposition 10** *If trial $t$ is a mistake, and $\mathbf{q}$ maximizes entropy subject to membership in $S_t \cap T_t$, then it is on the separating hyperplane for $T_t$.*

**Proof**: Because $\mathbf{q}$ and $\mathbf{p}$ both satisfy $S_t$, any convex combination of the two satisfies $S_t$. Thus, if $\mathbf{q}$ was on the interior of $T_t$, we could find a probability distribution with higher entropy that still satisfies both $S_t$ and $T_t$ by taking a tiny step from $\mathbf{q}$ toward $\mathbf{p}$. This would contradict the assumption that $\mathbf{q}$ is the maximum entropy member of $S_t \cap T_t$. ∎

This implies that the next hypothesis of a ROME algorithm is either on $J_t$ (the separating hyperplane $T_t$) only, or on both $J_t$ and $H_t$ (the separating hyperplane of $S_t$). The following theorem will enable us to obtain a formula in either case.

**Lemma 11 ([9] (Theorem 3.1))** *Suppose $\mathbf{q}$ is a probability distribution, and $C$ is a set defined by linear constraints as follows: for an $m \times n$ real matrix $A$, and a $m$-dimensional*

*column vector* $\mathbf{b}$, $C = \{\mathbf{r} : A\mathbf{r} = \mathbf{b}\}$. *Then if* $\mathbf{r}$ *is the member of* $C$ *minimizing* $D(\mathbf{r}||\mathbf{q})$, *then there are scalar constants* $Z, c_1, ..., c_m$ *such that for all* $i \in \{1, ..., n\}$, $r_i = \exp(\sum_{j=1}^m c_j a_{j,i})q_i/Z$.

If the next hypothesis $\mathbf{p}_{t+1}$ of a ROME algorithm is on $H_t$, then by Lemma 2, it and all other members of $H_t$ satisfy $D(\mathbf{p}_{t+1}||\mathbf{u}) = D(\mathbf{p}_{t+1}||\mathbf{p}_t) + D(\mathbf{p}_t||\mathbf{u})$. Thus, in this case, $\mathbf{p}_{t+1}$ also minimizes $D(\mathbf{q}||\mathbf{p}_t)$ from among the members $\mathbf{q}$ of $H_t \cap J_t$. Thus, Lemma 11 implies that $p_{t+1,i}/p_{t,i}$ is the same for all $i$ with $x_i = 1$, and the same for all $i$ with $x_i = 0$. This implies that, for $\mathrm{ROME}_{b,\gamma_+,\gamma_-}$, if there was a mistake on a trial $t$,

$$
p_{t+1,i} = \begin{cases}
\frac{(b+\gamma_+)p_{t,i}}{\mathbf{p}_t \cdot \mathbf{x}_t} & \text{if } x_{t,i} = 1 \text{ and } y_t = 1 \\
\frac{(1-(b+\gamma_+))p_{t,i}}{1-(\mathbf{p}_t \cdot \mathbf{x}_t)} & \text{if } x_{t,i} = 0 \text{ and } y_t = 1 \\
\frac{(b-\gamma_-)p_{t,i}}{\mathbf{p}_t \cdot \mathbf{x}_t} & \text{if } x_{t,i} = 1 \text{ and } y_t = 0 \\
\frac{(1-(b-\gamma_+))p_{t,i}}{1-(\mathbf{p}_t \cdot \mathbf{x}_t)} & \text{if } x_{t,i} = 0 \text{ and } y_t = 0.
\end{cases}
\tag{4}
$$

Note that this updates the weights multiplicatively, like Winnow and Weighted Majority.

If $\mathbf{p}_{t+1}$ is not on the separating hyperplane for $S_t$, then it must maximize entropy subject to membership in $T_t$ alone, and therefore subject to membership in $J_t$. In this case, Lemma 11 implies

$$
p_{t+1,i} = \begin{cases}
\frac{(b+\gamma_+)}{|\{j:x_{t,j}=1\}|} & \text{if } x_{t,i} = 1 \text{ and } y_t = 1 \\
\frac{(1-(b+\gamma_+))}{|\{j:x_{t,j}=0\}|.} & \text{if } x_{t,i} = 0 \text{ and } y_t = 1 \\
\frac{(b-\gamma_+)}{|\{j:x_{t,j}=1\}|} & \text{if } x_{t,i} = 1 \text{ and } y_t = 0 \\
\frac{(1-(b-\gamma_+))}{|\{j:x_{t,j}=0\}|.} & \text{if } x_{t,i} = 0 \text{ and } y_t = 0
\end{cases}
\tag{5}
$$

If this is the case, then $\mathbf{p}_{t+1}$ defined as in (5) should be a member of $S_t$.

How to test for membership in $S_t$? Evaluating the gradient of $H$ at $\mathbf{p}_t$, and simplifying a bit, we can see that

$$
S_t = \left\{ \mathbf{q} : \sum_{i=1}^n q_i \ln \frac{1}{p_{t,i}} \leq H(\mathbf{p}) \right\}.
$$

Summing up, a way to implement a ROME algorithm with the same mistake bound as the corresponding OME algorithm is to

- try defining $\mathbf{p}_{t+1}$ as in (5), and check whether the resulting $\mathbf{p}_{t+1} \in S_t$, if so use it, and

- if not, then define $\mathbf{p}_{t+1}$ as in (4) instead.

## Acknowledgements

We are grateful to Tony Jebara and Tong Zhang for helpful conversations, and an anonymous referee for suggesting a simplification of the proof of Theorem 3.

## References

[1] D. Angluin and L. Valiant. Fast probabilistic algorithms for Hamiltonion circuits and matchings. *Journal of Computer and System Sciences*, 18(2):155–193, 1979.

[2] A. L. Berger, S. Della Pietra, and V. J. Della Pietra. A maximum entropy approach to natural language processing. *Computational Linguistics*, 22(1):39–71, 1996.

[3] D. Blackwell. An analog of the minimax theorem for vector payoffs. *Pac. J. Math.*, 6:1–8, 1956.

[4] A. Blum, 2002. http://www-2.cs.cmu.edu/∼avrim/ML02/lect0418.txt.

[5] N. Cesa-Bianchi, A. Krogh, and M. Warmuth. Bounds on approximate steepest descent for likelihood maximization in exponential families. *IEEE Transactions on Information Theory*, 40(4):1215–1218, 1994.

[6] T. Cover and J. Thomas. *Elements of Information Theory*. Wiley, 1991.

[7] K. Crammer, O. Dekel, S. Shalev-Shwartz, and Y. Singer. Online passive-aggressive algorithms. *NIPS*, 2003.

[8] Koby Crammer and Yoram Singer. Ultraconservative online algorithms for multiclass problems. In *COLT*, pages 99–115, 2001.

[9] I. Csiszár. I-divergence geometry of probability distributions and minimization problems. *Annals of Probability*, 3:146–158, 1975.

[10] D. P. Dubhashi and A. Panconesi. Concentration of measure for the analysis of randomized algorithms, 1998. Monograph.

[11] Geoffrey J. Gordon. Regret bounds for prediction problems. In *Proc. 12th Annu. Conf. on Comput. Learning Theory*, pages 29–40. ACM Press, New York, NY, 1999.

[12] D. P. Helmbold, N. Littlestone, and P. M. Long. On-line learning with linear loss constraints. *Information and Computation*, 161(2):140–171, 2000.

[13] M. Herbster and M. K. Warmuth. Tracking the best linear predictor. *Journal of Machine Learning Research*, 1:281–309, 2001.

[14] T. Jaakkola, M. Meila, and T. Jebara. Maximum entropy discrimination. *NIPS*, 1999.

[15] J. Kivinen and M. Warmuth. Boosting as entropy projection. *COLT*, 1999.

[16] J. Kivinen and M. K. Warmuth. Additive versus exponentiated gradient updates for linear prediction. *Information and Computation*, 132(1):1–63, 1997.

[17] J. Langford, M. Seeger, and N. Megiddo. An improved predictive accuracy bound for averaging classifiers. *ICML*, pages 290–297, 2001.

[18] Y. Li and P. M. Long. The relaxed online maximum margin algorithm. *Machine Learning*, 46(1-3):361–387, 2002.

[19] N. Littlestone. *Mistake Bounds and Logarithmic Linear-threshold Learning Algorithms*. PhD thesis, UC Santa Cruz, 1989.

[20] N. Littlestone, P. M. Long, and M. K. Warmuth. On-line learning of linear functions. *Computational Complexity*, 5:1–23, 1995. Preliminary version in STOC'91.

[21] N. Littlestone and M. K. Warmuth. The weighted majority algorithm. *Information and Computation*, 108:212–261, 1994.

[22] A. B. J. Novikoff. On convergence proofs on perceptrons. In *Proceedings of the Symposium on the Mathematical Theory of Automata*, pages 615–622, 1962.

[23] M. S. Pinsker. *Information and Information Stability of Random Variables and Processes*. Holden-Day, 1964.

[24] F. Topsoe. Some inequalities for information divergence and related measures of discrimination. *IEEE Trans. Inform. Theory*, 46(4):1602–1609, 2001.

[25] P. Vaidya. A new algorithm for minimizing convex functions over convex sets. *FOCS*, pages 338–343, 1989.

[26] T. Zhang. Regularized winnow methods. *NIPS*, pages 703–709, 2000.

[27] T. Zhang. A sequential approximation bound for some sample-dependent convex optimization problems with applications in learning. *COLT*, pages 65–81, 2001.
